# MODELING THE OLFACTORY BULB
# — COUPLED NONLINEAR OSCILLATORS

Zhaoping Li†      J. J. Hopfield*
† Division of Physics, Mathematics and Astronomy
*Division of Biology, and Division of Chemistry and Chemical Engineering
†*California Institute of Technology, Pasadena, CA 91125, USA
*AT&T Bell Laboratories

## ABSTRACT

The olfactory bulb of mammals aids in the discrimination of odors. A mathematical model based on the bulbar anatomy and electrophysiology is described. Simulations produce a 35-60 Hz modulated activity coherent across the bulb, mimicking the observed field potentials. The decision states (for the odor information ) here can be thought of as stable cycles, rather than point stable states typical of simpler neuro-computing models. Analysis and simulations show that a group of coupled non-linear oscillators are responsible for the oscillatory activities determined by the odor input, and that the bulb, with appropriate inputs from higher centers, can enhance or suppress the sensitivity to particular odors. The model provides a framework in which to understand the transform between odor input and the bulbar output to olfactory cortex.

## 1. INTRODUCTION

The olfactory system has a simple cortical intrinsic structure (Shepherd 1979), and thus is an ideal candidate to yield insight on the principles of sensory information processing. It includes the receptor cells, the olfactory bulb, and the olfactory cortex receiving inputs from the bulb (Figure [1]). Both the bulb and the cortex exhibit similar 35-90 Hz rhythmic population activity modulated by breathing. Efforts have been made to model the bulbar information processing function (Freeman 1979b, 1979c; Freeman and Schneider 1982; Freeman and Skarda 1985; Baird 1986; Skarda and Freeman 1987), which is still unclear (Scott 1986). The bulbar position in the olfactory pathway and the linkage of the oscillatory activity with the sniff cycles suggest that the bulb and the oscillation play important roles in the olfactory information processing. We will examine how the bulbar oscillation pattern, which can be thought of as the decision state about odor information, originates and how it depends on the input odor. We then show that with appropriate inputs from the higher centers, the bulb can suppress or enhance the its sensitivity to particular odors. Much more details of our work are described in other two papers (Li and Hopfield 1988a, 1988b).

The olfactory bulb has mainly the excitatory mitral and the inhibitory granule cells located on different parallel lamina. Odor receptors effectively synapse on the mitral cells which interact locally with the granule cells and carry the bulbar outputs (Fig 1, Shepherd 1979). A rabbit has about 50, 000 mitral, and $\sim$ 10,000,000 granule cells (Shepherd 1979). With short odor pulses, the receptor firing rate increases in time, and terminates quickly after the odor pulse terminates (Getchell and Shepherd 1978). Most inputs from higher brain centers are directed to the granule cells, and little is know about them. The surface EEG wave (generated by granule activities, Freeman 1978; Freeman and Schneider 1982), depending on odor stimulations and animal motivation, shows a high amplitude oscillation arising during the inhalation and stopping early in the exhalation. The oscillation is an intrinsic property of the bulb itself, and is influenced by central inputs (Freeman 1979a; Freeman and Skarda 1985). It has a peak frequency (which is the same across the bulb) in the range of 35-90 Hz, and rides on a slow background wave phase locked with the respiratory wave.

## 2. MODEL ORGANIZATION

For simplicity, we only include ( $N$ excitatory ) mitral and ( $M$ inhibitory ) granule cells in the model. The Receptor input $I$ is $I_i \equiv I_{odor,i} + I_{background,i}$, for $1 \ldots, N$, a superposition of an odor signal $I_{odor}$ and a background input $I_{background}$. $I_{odor} \geq 0$ increases in time during inhalation, and return exponentially during exhalation toward the ambient. The central input to the granule cells is vector $I_c$ with components $I_{c,j}$ for $1 \leq j \leq M$. For now, it is assumed that $I_c = 0.1$ and $I_{background} = 0.243$ do not change during a sniff (Li and Hopfield 1988a).

Each cell is one unit with its internal state level described by a single variable, and its output a continuous function of the internal state level. The internal states and outputs are respectively $X = \{x_1, x_2, \ldots, x_N\}$ and $G_x(X) = \{g_x(x_1), g_x(x_2), \ldots, g_x(x_N)\}$ ($Y = \{y_1, y_2, \ldots, y_M\}$ and $G_y(Y) = \{g_y(y_1), g_y(y_2), \ldots, g_y(y_M)\}$) for the mitral (granule) cells, where $g_x \geq 0$ and $g_y \geq 0$ are the neurons' non-linear sigmoid output functions essential for the bulbar oscillation dynamics (Freeman and Skarda 1985) to be studied.

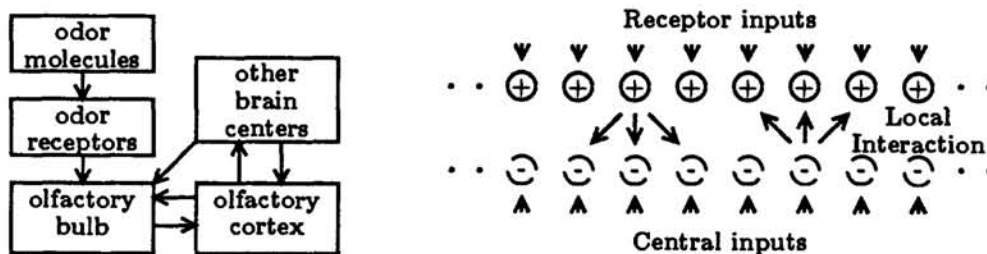

Fig.1. Left: olfactory system; Right: bulbar structure
Cells marked "+" are mitral cells, "-" are granule cells

The geometry of bulbar structure is simplified to a one dimensional ring. Each cell is specified by an index, e.g. $i^{th}$ mitral cell, and $j^{th}$ granule cell for all $i, j$

indicating cell locations on the ring (Fig 1). $N \times M$ matrix $H_o$ and $M \times N$ matrix $W_o$ are used respectively to describe the synaptic strengths (postsynaptic input : presynaptic output) from granule cells to mitral cells and vice versa. The bulb model system has equations of motion:

$$\dot{X} = -H_o G_y(Y) - \alpha_x X + I,$$
$$\dot{Y} = W_o G_x(X) - \alpha_y Y + I_c. \quad (2.1)$$

where $\alpha_x = 1/\tau_x$, $\alpha_y = 1/\tau_y$, and $\tau_x = \tau_y = 7\ msec$ are the time constants of the mitral and granule cells respectively (Freeman and Skarda 1985; Shepherd 1988). In simulation, weak random noise is added to $I$ and $I_c$ to simulate the fluctuations in the system.

## 3. SIMULATION RESULT

Computer simulation was done with 10 mitral and granule cells, and show that the model can capture the major effects of the real bulb. The rise and fall of oscillations with input and the baseline shift wave phase locked with sniff cycles are obvious (Fig.2). The simulated EEG (calculated using the approximation by Freeman (1980)) and the measured EEG are shown for comparison. During a sniff, all the cells oscillate coherently with the same frequency as physiologically observed.

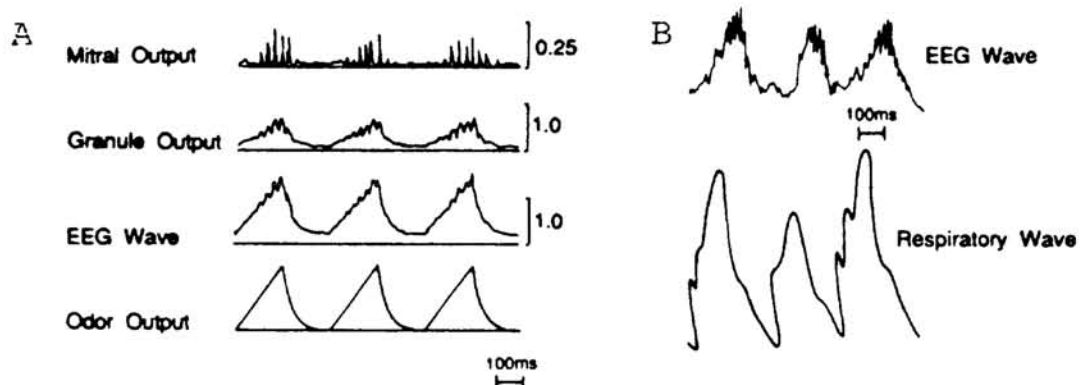

Fig.2. A: Simulation result; B: measured result from Freeman and Schneider 1982.

The model also shows the capability of a pattern classifier. During a sniff, some input patterns induce oscillation, while others do not, and different inputs induce different oscillation patterns. We showed (Li and Hopfield 1988a) that the bulb amplifies the differences between the different inputs to give different output patterns, while the responses to same odor inputs with different noise samples differ negligibly.

## 4. MATHEMATICAL ANALYSIS

A (damped) oscillator with frequency $\omega$ can be described by the equations

$$\dot{x} = -\omega y - \alpha x$$
$$\dot{y} = \omega x - \alpha y \qquad or \quad \ddot{x} + 2\alpha\dot{x} + (\omega^2 + \alpha^2)x = 0 \qquad (4.1)$$

The solution orbit in $(x, y)$ space is a circle if $\alpha = 0$ (non-damped oscillator), and spirals into the origin otherwise (damped oscillator). If a mitral cell and a granule cell are connected to each other, with inputs $i(t)$ and $i_c(t)$ respectively, then

$$
\begin{aligned}
\dot{x} &= -h \cdot g_y(y) - \alpha_x x + i(t), \\
\dot{y} &= w \cdot g_x(x) - \alpha_y y + i_c(t).
\end{aligned}
\tag{4.2}
$$

This is the scalar version of equation (2.1) with each upper case letter representing a vector or matrix replaced by a lower case letter representing a scalar. It is assumed that $i(t)$ has a much slower time course than $x$ or $y$ (frequency of sniffs $\ll$ characteristic neural oscillation frequency). Use the adiabatic approximation, and define the equilibrium point $(x_o, y_o)$ as

$$
\begin{aligned}
\dot{x}_o &\approx 0 = -h \cdot g_y(y_o) - \alpha_x x_o + i, \\
\dot{y}_o &\approx 0 = w \cdot g_x(x_o) - \alpha_y y_o + i_c.
\end{aligned}
\tag{4.3}
$$

Define $x' \equiv x - x_o$, $y' \equiv y - y_o$. Then

$$
\begin{aligned}
\dot{x}' &= -h(g_y(y) - g_y(y_o)) - \alpha_x x', \\
\dot{y}' &= w(g_x(x) - g_x(x_o)) - \alpha_y y'.
\end{aligned}
$$

(cf. equation (4.1)). If $\alpha_x = \alpha_y = 0$, then the solution orbit

$$
R \equiv \int_{x_o}^{x_o+x'} w(g_x(s) - g_x(x_o))ds + \int_{y_o}^{y_o+y'} h(g_y(s) - g_y(y_o))ds = constant
$$

is a closed curve in the original $(x, y)$ space surrounding the point $(x_o, y_o)$, i.e., $(x, y)$ oscillates around the point $(x_o, y_o)$. When the dissipation is included, $dR/dt < 0$, the orbit in $(x, y)$ space will spiral into the point $(x_o, y_o)$. Thus a connected pair of mitral and granule cells behaves as a damped non-linear oscillator, whose oscillation center $(x_o, y_o)$ is determined by the external inputs $i$ and $i_c$. For small oscillation amplitudes, it can be approximated by a sinusoidal oscillator via linearization around the $(x_o, y_o)$:

$$
\begin{aligned}
\dot{x} &= -h \cdot g_y'(y_o)y - \alpha_x x \\
\dot{y} &= w \cdot g_x'(x_o)x - \alpha_y y
\end{aligned}
\tag{4.4}
$$

where $(x, y)$ is the deviation from $(x_o, y_o)$. The solution is $x = r_o e^{-\alpha t} sin(\omega t + \phi)$ where $\alpha = (\alpha_x + \alpha_y)/2$ and $\omega = \sqrt{hwg_x'(x_o)g_y'(y_o) + (\alpha_x - \alpha_y)^2/4}$. If $\alpha_x = \alpha_y$, which is about right in the bulb, $\omega = \sqrt{hwg_x'(x_o)g_y'(y_o)}$. For the bulb, $\alpha \approx 0.3\omega$. The oscillation frequency depends on the synaptic strengths $h$ and $w$, and is modulated by the receptor and central input via $(x_o, y_o)$.

$N$ such mitral-granule pairs with cell interconnections between the pairs represent a group of $N$ coupled non-linear damped oscillators. This is exactly the situation in the olfactory bulb. The locality of synaptic connections in the bulb implies that the oscillator coupling is also local. (That there are many more granule cells than mitral cells only means that there is more than one granule cell in each oscillator.) Corresponding to equation (4.2) and (4.4), we have equation (2.1) and

$$\dot{X} = -H_o G'_y(Y_o)Y - \alpha_z X \equiv -HY - \alpha_z X,$$
$$\dot{Y} = W_o G'_z(X_o)X - \alpha_y Y \equiv WX - \alpha_y Y. \tag{4.5}$$

where $(X, Y)$ are now deviations from $(X_o, Y_o)$ and $G'_z(X_o)$ and $G'_y(Y_o)$ are diagonal matrices with elements: $[G'_z(X_o)]_{ii} = g'_z(x_{i,o}) \geq 0$, $[G'_y(Y_o)]_{jj} = g'_y(y_{j,o}) \geq 0$, for all $i, j$. Eliminating $Y$,

$$\ddot{X} + (\alpha_z + \alpha_y)\dot{X} + (A + \alpha_z \alpha_y)X = 0 \tag{4.6}$$

where $A \equiv HW = H_o G'_y(Y_o)W_o G'_z(X_o)$. The $i^{th}$ oscillator (mitral cell) follows the equation

$$\ddot{x}_i + (\alpha_z + \alpha_y)\dot{x}_i + (A_{ii} + \alpha_z \alpha_y)x_i + \sum_{j \neq i} A_{ij} x_j = 0 \tag{4.7}$$

(cf.equation (4.1)), the the last term describes the coupling between oscillators. Non-linear effect occurs when the amplitude is large, and make the oscillation wave form non-sinusoidal.

If $X_k$ is one of the eigenvectors of $A$ with eigenvalue $\lambda_k$, equation (4.6) has $k^{th}$ oscillation mode

$$X \propto X_k e^{i\omega_k t} \equiv X_k exp(-\frac{(\alpha_z + \alpha_y)}{2}t \pm i\sqrt{\lambda_k + \frac{(\alpha_z - \alpha_y)^2}{4}}t) \tag{4.8}$$

Components of $X_k$ indicate oscillators' relative amplitudes and phases (for each $k = 1, 2, \ldots, N$ independent mode). For simplicity, we set $\alpha_z = \alpha_y = \alpha$, then $X \propto X_k e^{-\alpha t \pm i\sqrt{\lambda_k}t}$. Each mode has frequency $Re\sqrt{\lambda_k}$, where $Re$ means the real part of a complex number. If $Re(-\alpha \pm i\sqrt{\lambda_k}) > 0$ is satisfied for some $k$, then the amplitude of the $k^{th}$ mode will increase with time, i.e. growing oscillation. Starting from an initial condition of arbitrary small amplitudes in linear analysis, the mode with the fastest growing amplitude will dominate the output, and the whole bulb will oscillate in the same frequency as observed physiologically (Freeman 1978; Freeman and Schneider 1982) as well as in the simulation. With the non-linear effect, the strongest mode will suppress the others, and the final activity output will be a single "mode" in a non-linear regime.

Because of the coupling between the (damped) oscillators, the equilibrium point $(X_o, Y_o)$ of a group of oscillators is **no longer always stable** with the possibility of growing oscillation modes. $\lambda_k$ must be complex in order to have $k^{th}$ mode grow. For this, a necessary (but not sufficient) condition is that matrix $A$ is non-symmetric. Those inputs that make matrix $A$ less symmetric will more likely induce the oscillatory output and thus presumably be noticed by the following olfactory cortex (see Li and Hopfield 1988a for details).

The consequences (also observed physiologically) of our model are (Freeman 1975,1978; Freeman and Schneider 1982; Li and Hopfield 1988a): 1): local mitral cells' oscillation phase leads that of the local granule cells by a quarter cycle; 2): oscillations across the bulb have the same dominant frequency whose range possible should be narrow; 3): there should be a non-zero phase gradient field across the bulb; 4): the oscillation activity will rise during the inhale and fall at exhale, and rides on a slow background baseline shift wave phase locked with the sniff cycles.

This model of the olfactory bulb can be generalized to other masses of inter-acting excitatory and inhibitory cells such as those in olfactory cortex, neocortex and hippocampus (Shepherd 1979) etc. where there may be connections between the excitatory cells as well as the inhibitory cells (Li and Hopfield 1988a). Suppose that $B_o$ and $C_o$ are excitatory-to-excitatory and inhibitory-to-inhibitory connection matrices respectively, then equation (4.6) becomes

$$\ddot{X} + (\alpha_x - B + \alpha_y + C)\dot{X} + (A + (\alpha_x - B)(\alpha_y + C))X = 0 \qquad (4.9)$$

where $B \equiv B_o G'_x(X_o)$ and $C \equiv H C_o G'_y(Y_o) H^{-1}$.

## 5. COMPUTATIONS IN THE OLFACTORY BULB
Receptor input $I$ influences $(X_o, Y_o)$ as follows

$$
\begin{aligned}
dX_o &\approx (\alpha^2 + HW)^{-1}(\alpha dI + d\dot{I}) \\
dY_o &\approx (\alpha^2 + WH)^{-1}(W dI - \alpha H^{-1} d\dot{I})
\end{aligned}
\qquad (5.1)
$$

This is how the odor input determines the bulbar output. Increasing $I_{odor}$ not only raises the mean activity level $(X_o, Y_o)$ (and thus the gain $(G'_x(X_o), G'_y(Y_o))$), but also slowly changes the oscillation modes by structurally changing the oscillation equation (4.6) through matrix $A = H_o G'_y(Y_o) W_o G'_x(X_o)$. If $(X_o, Y_o)$ is raised to such an extent that $Re(-\alpha \pm i\sqrt{\lambda_k}) > 0$ is satisfied for some mode $k$, the equilibrium point $(X_o, Y_o)$ becomes unstable and this mode emerges with oscillatory bursts. Different oscillation modes that emerge are indicative of the different odor inputs controlling the system parameters $(X_o, Y_o)$, and can be thought of as the decision states reached for odor information, i.e., the oscillation pattern classifies odors. When $(X_o, Y_o)$ is very low (e.g. before inhale), all modes are damped, and only small amplitude oscillations occur, driven by noise and the weak time variation of the odor input. The absence of oscillation can be interpreted by higher processing

centers as the absence of an odor (Skarda and Freeman 1987). Detailed analysis shows how the bulb selectively responds (or not to respond) to certain input patterns (Li and Hopfield 1988a) by choosing the synaptic connections appropriately. This means the bulb can have non-uniform sensitivities to different odor receptor inputs and achieve better odor discriminations.

## 6. PERFORMANCE OPTIMIZATION IN THE BULB

We discussed (Li and Hopfield 1988a) how the olfactory bulb makes the least information contamination between sniffs and changes the motivation level for odor discrimination. We further postulate with our model that the bulb, with appropriate inputs from the higher centers, can enhance or suppress the sensitivity to particular odors (details in Li and Hopfield 1988b). When the central input $I_c$ is not fixed, it can control the bulbar output by shifting $(X_o, Y_o)$, just as the odor input $I$ can, equation (5.1) becomes:

$$
\begin{aligned}
dX_o &\approx (\alpha^2 + HW)^{-1}(\alpha dI + d\dot{I} - HdI_c + \alpha W^{-1}d\dot{I_c}) \\
dY_o &\approx (\alpha^2 + WH)^{-1}(W dI - \alpha H^{-1}d\dot{I} + \alpha dI_c + d\dot{I_c})
\end{aligned}
\tag{6.1}
$$

Suppose that $I_c \equiv I_{c,background} + I_{c,control}$ where $I_{c,control}$ is the control signal which changes during a sniff. Olfactory adaptation is achieved by having an $I_{c,control} = I_c^{cancel}$ which cancels the effect of $I_{odor}$ on $X_o$ — cancelling. This keeps the mitral cells baseline output $G_x(X_o)$ and gain $G'_x(X_o)$ low, and thus makes the oscillation output impossible as if no odor exists. We can then expect that reversing the sign of $I_c^{cancel}$ will cause the bulb to have an enhanced, instead of reduced (adapted), response to $I_{odor}$ — anti-cancelling, and achieve the olfactory enhancement. We can derive further phenomena such as recognizing an odor component in an odor mixture, cross-adaptation and cross-enhancement (Li and Hopfield 1988b). Computer simulations confirmed the expected results.

## 7. DISCUSSION

Our model of the olfactory bulb is a simplification of the known anatomy and physiology. The net of the mitral and granule cells simulates a group of coupled non-linear oscillators which are the sources of the rhythmic activities in the bulb. The coupling makes the oscillation coherent across the bulb surface for each sniff. The model suggests, in agreement with Freeman and coworkers, that stability change bifurcation is used for the bulbar oscillator system to decide primitively on the relevance of the receptor input information. Different non-damping oscillation modes emerged are used to distinguish the different odor input information which is the driving source for the bifurcations, and are approximately thought of as the (unitary) decision states of the system for the odor information. With the extra information represented in the oscillation phases of the cells, the bulb emphasizes the differences between different input patterns (section 4). Both the analysis and simulation show that the bulb is selectively sensitive to different receptor input patterns. This selectivity as well as the motivation level of the animal could also be

modulated from higher centers. This model also successfully applies to bulbar ability to use input from higher centers to suppress or enhance sensitivity to particular target or to mask odors.

This model does not exclude the possibility that the information be coded in the non-oscillatory slow wave $X_0$ which is also determined by the odor input. The chief behaviors do not depend on the number of cells in the model. The model can be generalized to olfactory cortex, hippocampus and neocortex etc. where there are more varieties of synaptic organizations.

## Acknowledgements

This research was supported by ONR contract N00014-87-K-0377. We would also like to acknowledge discussions with J.A. Bower.

## References

Baird B. Nonlinear dynamics of pattern formation and pattern recognition in rabbit olfactory bulb. Physica **22D**, 150-175 (1986)

Freeman W.J. Mass action in the nervous system. New York: Academic Press 1975

Freeman W.J. Spatial properties of an EEG event in the olfactory bulb and cortex. Electroencephalogr. Clin. Neurophysiol. **44**, 586-605 (1978)

Freeman W.J. Nonlinear Gain mediating cortical stimulus-response relations. Biol. Cybernetics **33**, 237-247 (1979a)

Freeman W.J. Nonlinear dynamics of paleocortex manifested in the olfactory EEG. Biol. Cybernetics **35**, 21-37 (1979b)

Freeman W.J. EEG analysis gives model of neuronal template-matching mechanism for sensory search with olfactory bulb. Biol. Cybernetics **35**, 221-234 (1979c)

Freeman W.J. Use of spatial deconvolution to compensate for distortion of EEG by volume conduction. IEEE Trans. Biomed. Engineering **27**, 421-429 (1980)

Freeman W.J., Schneider W.S. Changes in spatial patterns of rabbit olfactory EEG with conditioning to odors. Psychophysiology **19**, 44-56 (1982)

Freeman W.J., Skarda C.A. Spatial EEG patterns, non-linear dynamics and perception: the Neo-Sherringtonian view. Brain Res. Rev. **10**, 147-175 (1985)

Getchell T.V., Shepherd G.M. Responses of olfactory receptor cells to step pulses of odour at different concentrations in the salamender. J. Physiol. **282**, 521-540 (1978)

Lancet D. Vertebrate olfactory reception Ann. Rev. Neurosci. **9**, 329-355 (1986)

Li Z., Hopfield J.J. Modeling the olfactory bulb. Submitted to Biological Cybernetics (1988a)

Li Z., Hopfield J.J. A model of olfactory adaptation and enhancement in the olfactory bulb. In preparation. (1988b)

Scott J.W. The olfactory bulb and central pathways. Experientia **42**, 223-232 (1986)

Shepherd G.M. The synaptic organization of the brain. New York: Oxford University Press 1979

Shepherd G.M. Private communications. (1988)

Skarda C.A., Freeman W.J. How brains make chaos in order to make sense of the world. Behavioral and Brain Sciences **10**, 161-195 (1987)